# Wavelet Models for Video Time-Series

**Sheng Ma and Chuanyi Ji**
Department of Electrical, Computer, and Systems Engineering
Rensselaer Polytechnic Institute, Troy, NY 12180
e-mail: shengm@ecse.rpi.edu, chuanyi@ecse.rpi.edu

## Abstract

In this work, we tackle the problem of time-series modeling of video traffic. Different from the existing methods which model the time-series in the time domain, we model the wavelet coefficients in the wavelet domain. The strength of the wavelet model includes (1) a unified approach to model both the long-range and the short-range dependence in the video traffic simultaneously, (2) a computationally efficient method on developing the model and generating high quality video traffic, and (3) feasibility of performance analysis using the model.

## 1 Introduction

As multi-media (compressed Variable Bit Rate (VBR) video, data and voice) traffic is expected to be the main loading component in future communication networks, accurate modeling of the multi-media traffic is crucial to many important applications such as video-conferencing and video-on-demand. From modeling standpoint, multi-media traffic can be regarded as a time-series, which can in principle be modeled by techniques in time-seres modeling. Modeling such a time-series, however, turns out to be difficult, since it has been found recently that real-time video and Ethernet traffic possesses the complicated temporal behavior which fails to be modeled by conventional methods[3][4]. One of the significant statistical properties found recently on VBR video traffic is the co-existence of the long-range (LRD) and the short-range (SRD) dependence (see for example [4][6] and references therein). Intuitively, this property results from scene changes, and suggests a complex behavior of video traffic in the time domain[7]. This complex temporal behavior makes accurate modeling of video traffic a challenging task. The goal of this work is to develop a unified and computationally efficient method to model both the long-range and the short-range dependence in real video sources.

Ideally, a good traffic model needs to be (*a*) accurate enough to characterize pertinent statistical properties in the traffic, (*b*) computationally efficient, and (*c*) fea-

sible for the analysis needed for network design. The existing models developed to capture both the long-range and the short-range dependence include Fractional Auto-regressive Integrated Moving Average (FARIMA) models[4], a model based on Hosking's procedure[6], Transform-Expand-Sample (TES) model[9] and scene-based models[7]. All these methods model both LRD and SRD in the time domain. The scene-based modeling[7] provides a physically interpretable model feasible for analysis but difficult to be made very accurate. TES method is reasonably fast but too complex for the analysis. The rest of the methods suffer from computational complexity too high to be used to generate a large volume of synthesized video traffic.

To circumvent these problems, we will model the video traffic in the wavelet domain rather than in the time domain. Motivated by the previous work on wavelet representations of (the LRD alone) Fractional Gaussian Noise (FGN) process (see [2] and references therein), we will show in this paper simple wavelet models can simultaneously capture the short-range and the long-rage dependence through modeling two video traces. Intuitively, this is due to the fact that the (deterministic) similar structure of wavelets provides a natural match to the (statistical) self-similarity of the long-range dependence. Then wavelet coefficients at each time scale is modeled based on simple statistics. Since wavelet transforms and inverse transforms is in the order of $O(N)$, our approach will be able to attain the lowest computational complexity to generate wavelet models. Furthermore, through our theoretical analysis on the buffer loss rate, we will also demonstrate the feasibility of using wavelet models for theoretical analysis.

## 1.1  Wavelet Transforms

In $L^2(R)$ space, discrete wavelets $\phi_j^m(t)$'s are ortho-normal basis which can be represented as $\phi_j^m(t) = 2^{-j/2}\phi(2^{-j}t - m)$, for $t \in [0, 2^K - 1]$ with $K \geq 1$ being an integer. $\phi(t)$ is the so-called mother wavelet. $1 \leq j \leq K$ and $0 \leq m \leq 2^{K-j} - 1$ represent the time-scale and the time-shift, respectively. Since wavelets are the dilation and shift of a mother wavelet, they possess a deterministic similar structure at different time scales. For simplicity, the mother wavelet in this work is chosen to be the Haar wavelet, where $\phi(t)$ is 1 for $0 \leq t < 1/2$, -1 for $1/2 \leq t < 1$ and 0 otherwise.

Let $d_j^m$'s be wavelet coefficients of a discrete-time process $x(t)$ ($t \in [0, 2^K - 1]$). Then $d_j^m$ can be obtained through the wavelet transform $d_j^m = \sum_{t=0}^{2^K - 1} x(t)\phi_j^m(t)$. $x(t)$ can be represented through the inverse wavelet transform $x(t) = \sum_{j=1}^{K} \sum_{m=0}^{2^{K-j} - 1} d_j^m \phi_j^m(t) + \phi_0$, where $\phi_0$ is equal to the average of $x(t)$.

## 2  Wavelet Modeling of Video Traffic

### 2.1  The Video Sources

Two video sources are used to test our wavelet models: (1) "Star Wars"[4], where each frame is encoded by JPEG-like encoder, and (2) MPEG coded videos at Group of Pictures (GOP) level[7][11] called "MPEG GOP" in the rest of the paper. The modeling is done at either the frame level or the GOP level.

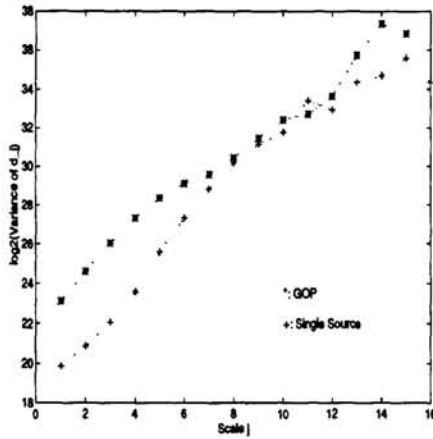

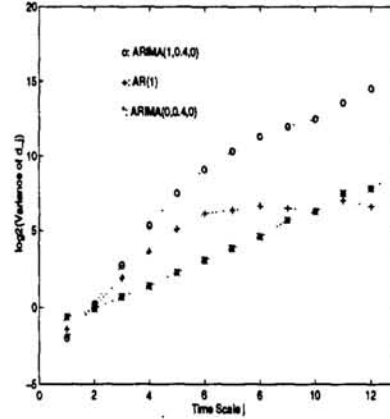

Figure 1: Log 2 of Variance of $d_j$ versus the time scale $j$

Figure 2: Log 2 of Variance of $d_j$ versus the time scale $j$

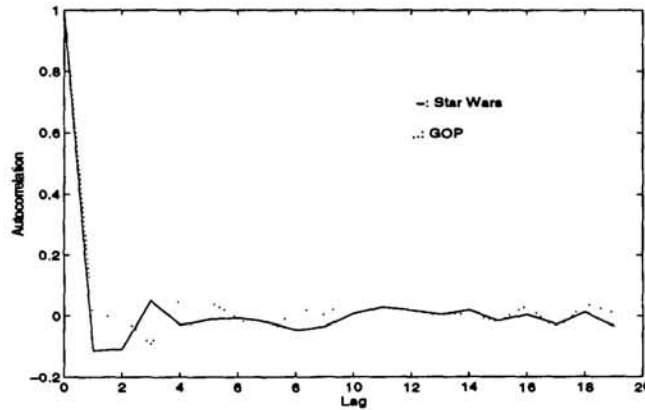

Figure 3: The sample autocorrelations of $d_6^m$.

## 2.2 The Variances and Auto-correlation of Wavelet Coefficients

As the first step to understand how wavelets capture the LRD and SRD, we plot in Figure (1) the variance of the wavelet coefficients $d_j^m$'s at different time scales for both sources. To understand what the curves mean, we also plot in Figure (2) the variances of wavelet coefficients for three well-known processes: FARIMA$(0, 0.4, 0)$, FARIMA$(1, 0.4, 0)$, and AR$(1)$. FARIMA$(0, 0.4, 0)$ is a long-range dependent process with Hurst parameter $H = 0.9$. AR$(1)$ is a short-range dependent process, and FARIMA$(1, 0.4, 0)$ is a mixture of the long-range and the short-range dependent process.

As observed, for FARIMA$(0, 0.4, 0)$ process (LRD alone), the variance increases with $j$ exponentially for all $j$. For AR$(1)$ (SRD alone), the variance increases at an even faster rate than that of FARIMA$(0, 0.4, 0)$ when $j$ is small but saturates when $j$ is large. For FARIMA$(1, 0.4, 0)$, the variance shows the mixed properties from both AR$(1)$ and FARIMA$(0, 0.4, 0)$. The variance of the video sources behaves similarly to that of FARIMA$(1, 0.4, 0)$, and thus demonstrate the co-existence of the SRD and LRD in the video sources in the wavelet domain.

Figure 3 gives the sample auto-correlation of $d_6^m$ in terms of $m$'s. The auto-correlation function of the wavelet coefficients approaches zero very rapidly, and

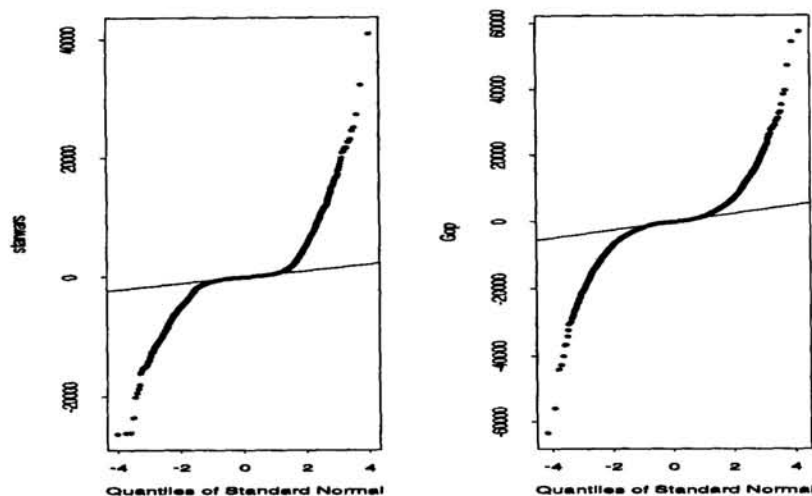

Figure 4: Quantile-Quantile of $d_j^m$ for $j = 3$. Left: Star Wars. Right: GOP.

thus indicates the short-range dependence in the wavelet domain. This suggests that although the autocorrelation of the video traffic is complex in the time-domain, modeling wavelet coefficients may be done using simple statistics within each time scale. Similar auto-correlations have been observed for the other $j$'s.

## 2.3 Marginal Probability Density Functions

Is variance sufficient for modeling wavelet coefficients? Figure (4) plots the $Q - Q$ plots for the wavelet coefficients of the two sources at $j = 3$[1]. The figure shows that the sample marginal density functions of wavelet coefficients for both the "Star Wars" and the MPEG GOP source at the given time scale have a much heavier tail than that of the normal distribution. Therefore, the variance alone is only sufficient when the marginal density function is normal, and in general a marginal density function should be considered as another pertinent statistical property.

It should be noted that correlation among wavelet coefficients at different time scales is neglected in this work for simplicity. We will show both empirically and theoretically that good performance in terms of sample auto-correlation and sample buffer loss probability can be obtained by a corresponding simple algorithm. More careful treatment can be found in [8].

## 2.4 An Algorithm for Generating Wavelet Models

The algorithm we derive include three main steps: (a) obtain sample variances of wavelet coefficients at each time scale, (b) generate wavelet coefficients independently from the normal marginal density function using the sample mean and variance [2], and (c) perform a transformation on the wavelet coefficients so that the

resulting wavelet coefficients have a marginal density function required by the traffic. The obtained wavelet coefficients form a wavelet model from which synthesized video traffic can be generated. The algorithm can be summarized as follows.

Let $\hat{x}(t)$ be the video trace of length $N$.

## Algorithm

1. Obtain wavelet coefficients from $\hat{x}(t)$ through the wavelet transform.

2. Compute the sample variance $\hat{\sigma}_j$ of wavelet coefficients at each time scale $j$.

3. Generate new wavelet coefficients $d_j^m$'s for all $j$ and $m$ independently through Gaussian distributions with variances $\hat{\sigma}_j$'s obtained at the previous step.

4. Perform a transformation on the wavelet coefficients so that the marginal density function of wavelet coefficients is consistent with that determined by the video traffic(see [6] for details on the transformation).

5. Do inverse wavelet transform using the wavelet coefficients obtained at the previous step to get the synthesized video traffic in the time domain.

The computational complexity of both the wavelet transform (Step 1) and the inverse transform (Step 5) is $O(N)$. So is for Steps 2, 3 and 4. Then $O(N)$ is the computational cost of the algorithm, which is the lowest attainable for traffic models.

### 2.5   Experimental Results

Video traces of length $171,000$ for "Star Wars" and $66369$ for "MPEG GOP" are used to obtain wavelet models. FARIMA models with 45 parameters are also obtained using the same data for comparison. The synthesized video traffic from both models are generated and used to obtain sample auto-correlation functions in the time-domain, and to estimate the buffer loss rate. The results[3] are given in Figure (6). Wavelet models have shown to outperform the FARIMA model.

For the computation time, it takes more than 5-hour CPU time[4] on a SunSPARC 5 workstation to develop the FARIMA model and to generate synthesized video traffic of length $171,000$[5]. It only takes 3 minutes on the same machine for our algorithm to complete the same tasks.

## 3   Theory

It has been demonstrated empirically in the previous section that the wavelet model, which ignores the correlation among wavelet coefficients of a video trace, can match well the sample auto-correlation function and the buffer loss probability. To further evaluate the feasibility of the wavelet model, the buffer overflow probability has been analyzed theoretically in [8]. Our result can be summarized in the following theorem.

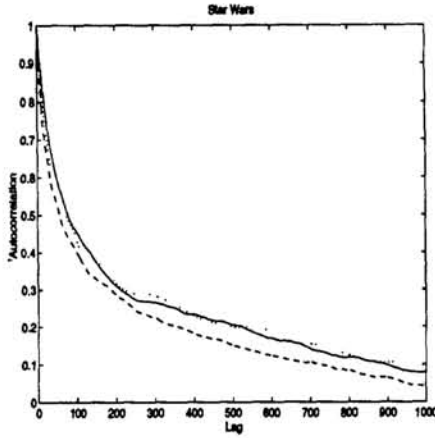

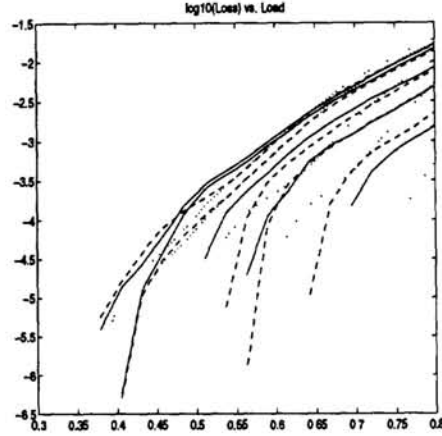

Figure 5: "-": Autocorrelation of "Star Wars"; "- -": ARIMA(25,d,20); "..": Our Algorithm

Figure 6: Loss rate attained via simulation. Vertical axis: $\log_{10}$(Loss Rate); horizontal axis: work load. "-": the single video source; "..": FARIMA(25,d,20); "–" Our algorithm. The normalized buffer size: 0.1, 1, 10,30 and 100 from the top down.

**Theorem** *Let $B_N$ and $\hat{B}_N$ be the buffer sizes at the $N$th time slot due to the synthesized traffic by the our wavelet model, and by the FGN process, respectively. Let $C$ and $B$ represent the capacity, and the maximum allowable buffer size respectively. Then*

$$
\begin{aligned}
\ln Pr(B_N > B) &\sim \ln Pr(\hat{B}_N > B) \\
&\sim -\frac{(C-\mu)^2(\frac{B}{C-\mu})^{2(1-H)}(\frac{1-H}{H})^{2H}}{2\sigma^2(1-H)^2},
\end{aligned} \tag{1}
$$

*where $\frac{1}{2} < H < 1$ is the Hurst parameter. $\mu$ and $\sigma^2$ is the mean and the variance of the traffic, respectively. $B$ is assume to be $(C-\mu)2^{k_0}$, where $k_0$ is a positive integer.*

This result demonstrates that using our simple wavelet model which neglects the correlations among wavelet coefficients, buffer overflow probability obtained is similar to that of the original FGN process as given in[10]. In other words, it shows that the wavelet model for a FGN process can have good modeling performance in terms of the buffer overflow criterion.

We would like to point out that the above theorem is held for a FGN process. Further work are needed to account for more general processes.

## 4 Conclusions

In this work, we have described an important application on time-series modeling: modeling video traffic. We have developed a wavelet model for the time-series. Through analyzing statistical properties of the time-series and comparing the wavelet model with FARIMA models, we show that one of the key factors to successfully model a time-series is to choose an appropriate model which naturally fits the pertinent statistical properties of the time-series. We have shown wavelets are particularly feasible for modeling the self-similar time-series due to the video traffic.

We have developed a simple algorithm for the wavelet models, and shown that the models are accurate, computationally efficient and simple enough for analysis.

## Footnotes

[1]Similar behaviors have been observed at the other time scales. A $Q - Q$ plot is a standard statistical tool to measure the deviation of a marginal density function from a normal density. The $Q - Q$ plots of a process with a normal marginal is a straight line. The deviation from the line indicates the deviation from the normal density. See [4] and references therein for more details.

[2]The mean of the wavelet coefficients can be shown to be zero for stationary processes.

[3]Due to page limit, we only provide plots for JPEG. GOP has similar results and was reported in [8].

[4]Computation time includes both parameter estimation and synthesized traffic generation.

[5]The computational complexity to generate synthesized video traffic of length $N$ is $O(N^2)$ for an FARIMA model[5][4].

## References

[1] I. Daubechies, *Ten Lectures on Wavelets*. Philadelphia: SIAM, 1992.

[2] Patrick Flandrin, "Wavelet Analysis and Synthesis of Fractional Brownian Motion", *IEEE transactions on Information Theory*, vol. 38, No.2, pp.910-917, 1992.

[3] W.E Leland, M.S. Taqqu, W. Willinger and D.V. Wilson, "On the Self-Similar Nature of Ethernet Traffic (Extended Version)", *IEEE/ACM Transactions on Networking*, vol.2, 1-14, 1994.

[4] Mark W. Garrett and Walter Willinger. "Analysis, Modeling and Generation of Self-Similar VBR Video Traffic.", in Proceedings of ACM SIGCOMM'94, London, U.K, Aug., 1994

[5] J.R.M. Hosking, "Modeling Persistence in Hydrological Time Series Using Fractional Differencing", *Water Resources Research*, 20, pp. 1898-1908, 1984.

[6] C. Huang, M. Devetsikiotis, I. Lambadaris and A.R. Kaye, "Modeling and Simulation of Self-Similar Variable Bit Rate Compressed Video: A Unified Approach", in Proceedings of ACM SIGCOMM'95, pp. 114-125.

[7] Predrag R. Jelenlnovic, Aurel A. Lazar, and Nemo Semret. The effect of multiple time scales and subexponentiality in mpeg video streams on queuing behavior. *IEEE Journal on Selected Area of Communications*, 15, to appear in May 1997.

[8] S. Ma and C. Ji, "Modeling Video Traffic in Wavelet Domain", to appear *IEEE INFOCOM*, 1998.

[9] B. Melamed, D. Raychaudhuri, B. Sengupta, and J. Zdepski. Tes-based video source modeling for performance evaluation of integrated networks. *IEEE Transactions on Communications*, 10, 1994.

[10] Ilkka Norros, "A storage model with self-similar input," *Queuing Systems*, vol.16, 387-396, 1994.

[11] O. Rose. "Statistical properties of mpeg video traffic and their impact on traffic modeling in atm traffic engineering", Technical Report 101, University of Wurzburg, 1995.